# Support Vector Method for Novelty Detection

**Bernhard Schölkopf\***, **Robert Williamson§**,
**Alex Smola§**, **John Shawe-Taylor†**, **John Platt\***

\* Microsoft Research Ltd., 1 Guildhall Street, Cambridge, UK
§ Department of Engineering, Australian National University, Canberra 0200
† Royal Holloway, University of London, Egham, UK
\* Microsoft, 1 Microsoft Way, Redmond, WA, USA
bsc/jplatt@microsoft.com, Bob.Williamson/Alex.Smola@anu.edu.au, john@dcs.rhbnc.ac.uk

## Abstract

Suppose you are given some dataset drawn from an underlying probability distribution $P$ and you want to estimate a "simple" subset $S$ of input space such that the probability that a test point drawn from $P$ lies outside of $S$ equals some a priori specified $\nu$ between 0 and 1.

We propose a method to approach this problem by trying to estimate a function $f$ which is positive on $S$ and negative on the complement. The functional form of $f$ is given by a kernel expansion in terms of a potentially small subset of the training data; it is regularized by controlling the length of the weight vector in an associated feature space. We provide a theoretical analysis of the statistical performance of our algorithm.

The algorithm is a natural extension of the support vector algorithm to the case of unlabelled data.

## 1 INTRODUCTION

During recent years, a new set of kernel techniques for supervised learning has been developed [8]. Specifically, support vector (SV) algorithms for pattern recognition, regression estimation and solution of inverse problems have received considerable attention. There have been a few attempts to transfer the idea of using kernels to compute inner products in feature spaces to the domain of *unsupervised* learning. The problems in that domain are, however, less precisely specified. Generally, they can be characterized as estimating *functions* of the data which tell you something interesting about the underlying distributions. For instance, kernel PCA can be characterized as computing functions which on the training data produce unit variance outputs while having minimum norm in feature space [4]. Another kernel-based unsupervised learning technique, regularized principal manifolds [6], computes functions which give a mapping onto a lower-dimensional manifold minimizing a regularized quantization error. Clustering algorithms are further examples of unsupervised learning techniques which can be kernelized [4].

An extreme point of view is that unsupervised learning is about estimating densities. Clearly, knowledge of the density of $P$ would then allow us to solve whatever problem can be solved on the basis of the data. The present work addresses an easier problem: it

proposes an algorithm which computes a binary function which is supposed to capture regions in input space where the probability density lives (its support), i.e. a function such that most of the data will live in the region where the function is nonzero [5]. In doing so, it is in line with Vapnik's principle never to solve a problem which is more general than the one we actually need to solve. Moreover, it is applicable also in cases where the density of the data's distribution is not even well-defined, e.g. if there are singular components. Part of the motivation for the present work was the paper [1]. It turns out that there is a considerable amount of prior work in the statistical literature; for a discussion, cf. the full version of the present paper [3].

## 2   ALGORITHMS

We first introduce terminology and notation conventions. We consider training data $\mathbf{x}_1, \ldots, \mathbf{x}_\ell \in \mathcal{X}$, where $\ell \in \mathbb{N}$ is the number of observations, and $\mathcal{X}$ is some set. For simplicity, we think of it as a compact subset of $\mathbb{R}^N$. Let $\Phi$ be a feature map $\mathcal{X} \to F$, i.e. a map into a dot product space $F$ such that the dot product in the image of $\Phi$ can be computed by evaluating some simple kernel [8]

$$k(\mathbf{x}, \mathbf{y}) = (\Phi(\mathbf{x}) \cdot \Phi(\mathbf{y})), \tag{1}$$

such as the Gaussian kernel

$$k(\mathbf{x}, \mathbf{y}) = e^{-\|\mathbf{x}-\mathbf{y}\|^2/c}. \tag{2}$$

Indices $i$ and $j$ are understood to range over $1, \ldots, \ell$ (in compact notation: $i, j \in [\ell]$). Bold face greek letters denote $\ell$-dimensional vectors whose components are labelled using normal face typeset.

In the remainder of this section, we shall develop an algorithm which returns a function $f$ that takes the value $+1$ in a "small" region capturing most of the data points, and $-1$ elsewhere. Our strategy is to map the data into the feature space corresponding to the kernel, and to separate them from the origin with maximum margin. For a new point $\mathbf{x}$, the value $f(\mathbf{x})$ is determined by evaluating which side of the hyperplane it falls on, in feature space. Via the freedom to utilize different types of kernel functions, this simple geometric picture corresponds to a variety of nonlinear estimators in input space.

To separate the data set from the origin, we solve the following quadratic program:

$$\min_{w \in F, \boldsymbol{\xi} \in \mathbb{R}^\ell, \rho \in \mathbb{R}} \quad \tfrac{1}{2}\|w\|^2 + \tfrac{1}{\nu\ell} \sum_i \xi_i - \rho \tag{3}$$

$$\text{subject to} \quad (w \cdot \Phi(\mathbf{x}_i)) \geq \rho - \xi_i, \ \xi_i \geq 0. \tag{4}$$

Here, $\nu \in (0, 1)$ is a parameter whose meaning will become clear later. Since nonzero slack variables $\xi_i$ are penalized in the objective function, we can expect that if $w$ and $\rho$ solve this problem, then the decision function $f(\mathbf{x}) = \text{sgn}((w \cdot \Phi(\mathbf{x})) - \rho)$ will be positive for most examples $\mathbf{x}_i$ contained in the training set, while the SV type regularization term $\|w\|$ will still be small. The actual trade-off between these two goals is controlled by $\nu$. Deriving the dual problem, and using (1), the solution can be shown to have an SV expansion

$$f(\mathbf{x}) = \text{sgn}\left(\sum_i \alpha_i k(\mathbf{x}_i, \mathbf{x}) - \rho\right) \tag{5}$$

(patterns $\mathbf{x}_i$ with nonzero $\alpha_i$ are called SVs), where the coefficients are found as the solution of the dual problem:

$$\min_{\boldsymbol{\alpha}} \frac{1}{2} \sum_{ij} \alpha_i \alpha_j k(\mathbf{x}_i, \mathbf{x}_j) \ \text{subject to} \ 0 \leq \alpha_i \leq \frac{1}{\nu\ell}, \ \sum_i \alpha_i = 1. \tag{6}$$

This problem can be solved with standard QP routines. It does, however, possess features that sets it apart from generic QPs, most notably the simplicity of the constraints. This can be exploited by applying a variant of SMO developed for this purpose [3].

The offset $\rho$ can be recovered by exploiting that for any $\alpha_i$ which is not at the upper or lower bound, the corresponding pattern $\mathbf{x}_i$ satisfies $\rho = (w \cdot \Phi(\mathbf{x}_i)) = \sum_j \alpha_j k(\mathbf{x}_j, \mathbf{x}_i)$.

Note that if $\nu$ approaches 0, the upper boundaries on the Lagrange multipliers tend to infinity, i.e. the second inequality constraint in (6) becomes void. The problem then resembles the corresponding *hard margin* algorithm, since the penalization of errors becomes infinite, as can be seen from the primal objective function (3). It can be shown that if the data set is separable from the origin, then this algorithm will find the unique supporting hyperplane with the properties that it separates all data from the origin, and its distance to the origin is maximal among all such hyperplanes [3]. If, on the other hand, $\nu$ approaches 1, then the constraints alone only allow one solution, that where all $\alpha_i$ are at the upper bound $1/(\nu\ell)$. In this case, for kernels with integral 1, such as normalized versions of (2), the decision function corresponds to a thresholded Parzen windows estimator.

To conclude this section, we note that one can also use *balls* to describe the data in feature space, close in spirit to the algorithms of [2], with hard boundaries, and [7], with "soft margins." For certain classes of kernels, such as Gaussian RBF ones, the corresponding algorithm can be shown to be equivalent to the above one [3].

## 3   THEORY

In this section, we show that the parameter $\nu$ characterizes the fractions of SVs and outliers (Proposition 1). Following that, we state a robustness result for the soft margin (Proposition 2) and error bounds (Theorem 5). Further results and proofs are reported in the full version of the present paper [3]. We will use italic letters to denote the feature space images of the corresponding patterns in input space, i.e. $x_i := \Phi(\mathbf{x}_i)$.

**Proposition 1** *Assume the solution of (4) satisfies $\rho \neq 0$. The following statements hold:*
*(i) $\nu$ is an upper bound on the fraction of outliers.*
*(ii) $\nu$ is a lower bound on the fraction of SVs.*
*(iii) Suppose the data were generated independently from a distribution $P(\mathbf{x})$ which does not contain discrete components. Suppose, moreover, that the kernel is analytic and nonconstant. With probability 1, asymptotically, $\nu$ equals both the fraction of SVs and the fraction of outliers.*

The proof is based on the constraints of the dual problem, using the fact that outliers must have Lagrange multipliers at the upper bound.

**Proposition 2** *Local movements of outliers parallel to $w$ do not change the hyperplane.*

We now move on to the subject of generalization. Our goal is to bound the probability that a novel point drawn from the same underlying distribution lies outside of the estimated region by a certain margin. We start by introducing a common tool for measuring the capacity of a class $\mathcal{F}$ of functions that map $\mathcal{X}$ to $\mathbb{R}$.

**Definition 3** *Let $(X, d)$ be a pseudo-metric space,[1] let $A$ be a subset of $X$ and $\epsilon > 0$. A set $B \subseteq X$ is an $\epsilon$-cover for $A$ if, for every $a \in A$, there exists $b \in B$ such that $d(a, b) \leq \epsilon$. The $\epsilon$-covering number of $A$, $\mathcal{N}_d(\epsilon, A)$, is the minimal cardinality of an $\epsilon$-cover for $A$ (if there is no such finite cover then it is defined to be $\infty$).*

The idea is that $B$ should be finite but approximate all of $A$ with respect to the pseudometric $d$. We will use the $l_\infty$ distance over a finite sample $X = (x_1, \ldots, x_\ell)$ for the pseudo-metric in the space of functions, $d_X(f, g) = \max_{i \in [\ell]} |f(x_i) - g(x_i)|$. Let $\mathcal{N}(\epsilon, \mathcal{F}, \ell) = \sup_{X \in \mathcal{X}^\ell} \mathcal{N}_{d_X}(\epsilon, \mathcal{F})$. Below, logarithms are to base 2.

**Theorem 4** *Consider any distribution $P$ on $\mathcal{X}$ and any $\theta \in \mathbb{R}$. Suppose $x_1, \ldots, x_\ell$ are generated i.i.d. from $P$. Then with probability $1 - \delta$ over such an $\ell$-sample, if we find $f \in \mathcal{F}$ such that $f(x_i) \geq \theta + \gamma$ for all $i \in [\ell]$,*

$$P\{x : f(x) < \theta - \gamma\} \leq \tfrac{2}{\ell}(k + \log \tfrac{2\ell}{\delta}),$$

*where $k = \lceil \log \mathcal{N}(\gamma, \mathcal{F}, 2\ell) \rceil$.*

We now consider the possibility that for a small number of points $f(x_i)$ fails to exceed $\theta + \gamma$. This corresponds to having a non-zero slack variable $\xi_i$ in the algorithm, where we take $\theta + \gamma = \rho/\|w\|$ and use the class of linear functions in feature space in the application of the theorem. There are well-known bounds for the log covering numbers of this class.

Let $f$ be a real valued function on a space $\mathcal{X}$. Fix $\theta \in \mathbb{R}$. For $x \in \mathcal{X}$, define

$$d(x, f, \gamma) = \max\{0, \theta + \gamma - f(x)\}.$$

Similarly for a training sequence $X$, we define $\mathcal{D}(X, f, \gamma) = \sum_{x \in X} d(x, f, \gamma)$.

**Theorem 5** *Fix $\theta \in \mathbb{R}$. Consider a fixed but unknown probability distribution $P$ on the input space $\mathcal{X}$ and a class of real valued functions $\mathcal{F}$ with range $[a, b]$. Then with probability $1 - \delta$ over randomly drawn training sequences $x$ of size $\ell$, for all $\gamma > 0$ and any $f \in \mathcal{F}$,*

$$P\{x : f(x) < \theta - \gamma \text{ and } x \notin X\} \leq \tfrac{2}{\ell}(k + \log \tfrac{4\ell}{\delta}),$$

*where $k = \left\lceil \log \mathcal{N}(\gamma/2, \mathcal{F}, 2\ell) + \frac{64(b-a)\mathcal{D}(X,f,\gamma)}{\gamma^2} \log \left( \frac{e\ell\gamma}{8\mathcal{D}(X,f,\gamma)} \right) \log \left( \frac{32\ell(b-a)^2}{\gamma^2} \right) \right\rceil$.*

The theorem bounds the probability of a new point falling in the region for which $f(x)$ has value less than $\theta - \gamma$, this being the complement of the estimate for the support of the distribution. The choice of $\gamma$ gives a trade-off between the size of the region over which the bound holds (increasing $\gamma$ increases the size of the region) and the size of the probability with which it holds (increasing $\gamma$ decreases the size of the log covering numbers).

The result shows that we can bound the probability of points falling outside the region of estimated support by a quantity involving the ratio of the log covering numbers (which can be bounded by the fat shattering dimension at scale proportional to $\gamma$) and the number of training examples, plus a factor involving the 1-norm of the slack variables. It is stronger than related results given by [1], since their bound involves the square root of the ratio of the Pollard dimension (the fat shattering dimension when $\gamma$ tends to 0) and the number of training examples.

The output of the algorithm described in Sec. 2 is a function $f(x) = \sum_i \alpha_i k(x_i, x)$ which is greater than or equal to $\rho - \xi_i$ on example $x_i$. Though non-linear in the input space, this function is in fact linear in the feature space defined by the kernel $k$. At the same time the 2-norm of the weight vector is given by $B = \sqrt{\alpha^T K \alpha}$, and so we can apply the theorem with the function class $\mathcal{F}$ being those linear functions in the feature space with 2-norm bounded by $B$. If we assume that $\theta$ is fixed, then $\gamma = \rho - \theta$, hence the support of the distribution is the set $\{x : f(x) \geq \theta - \gamma = 2\theta - \rho\}$, and the bound gives the probability of a randomly generated point falling outside this set, in terms of the log covering numbers of the function class $\mathcal{F}$ and the sum of the slack variables $\xi_i$. Since the log covering numbers

at scale $\gamma/2$ of the class $\mathcal{F}$ can be bounded by $O(\frac{R^2 B^2}{\gamma^2} \log^2 \ell)$ this gives a bound in terms of the 2-norm of the weight vector.

Ideally, one would like to allow $\theta$ to be chosen after the value of $\rho$ has been determined, perhaps as a fixed fraction of that value. This could be obtained by another level of structural risk minimisation over the possible values of $\rho$ or at least a mesh of some possible values. This result is beyond the scope of the current preliminary paper, but the form of the result would be similar to Theorem 5, with larger constants and log factors.

Whilst it is premature to give specific theoretical recommendations for practical use yet, one thing is clear from the above bound. To generalize to novel data, the decision function to be used should employ a threshold $\eta \cdot \rho$, where $\eta < 1$ (this corresponds to a nonzero $\gamma$).

## 4   EXPERIMENTS

We apply the method to artificial and real-world data. Figure 1 displays 2-D toy examples, and shows how the parameter settings influence the solution.

Next, we describe an experiment on the USPS dataset of handwritten digits. The database contains 9298 digit images of size $16 \times 16 = 256$; the last 2007 constitute the test set. We trained the algorithm, using a Gaussian kernel (2) of width $c = 0.5 \cdot 256$ (a common value for SVM classifiers on that data set, cf. [2]), on the test set and used it to identify outliers — it is folklore in the community that the USPS test set contains a number of patterns which are hard or impossible to classify, due to segmentation errors or mislabelling. In the experiment, we augmented the input patterns by ten extra dimensions corresponding to the class labels of the digits. The rationale for this is that if we disregarded the labels, there would be no hope to identify *mislabelled* patterns as outliers. Fig. 2 shows the 20 worst outliers for the USPS test set. Note that the algorithm indeed extracts patterns which are very hard to assign to their respective classes. In the experiment, which took 36 seconds on a Pentium II running at 450 MHz, we used a $\nu$ value of 5%.

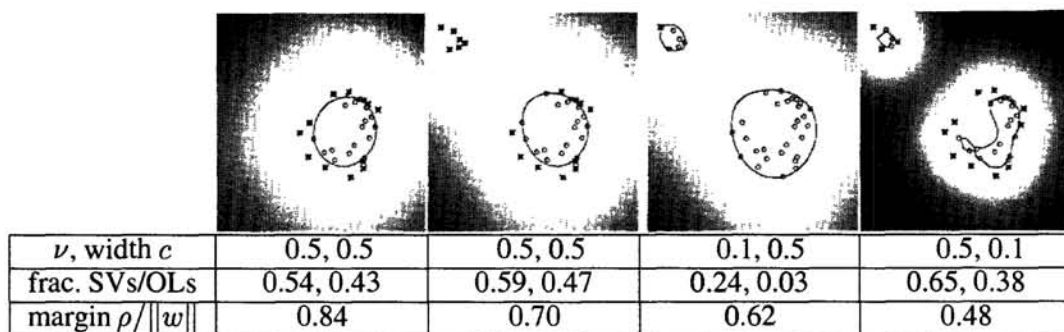

| $\nu$, width $c$ | 0.5, 0.5 | 0.5, 0.5 | 0.1, 0.5 | 0.5, 0.1 |
|---|---|---|---|---|
| frac. SVs/OLs | 0.54, 0.43 | 0.59, 0.47 | 0.24, 0.03 | 0.65, 0.38 |
| margin $\rho/\|w\|$ | 0.84 | 0.70 | 0.62 | 0.48 |

Figure 1: *First two pictures:* A single-class SVM applied to two toy problems; $\nu = c = 0.5$, domain: $[-1, 1]^2$. Note how in both cases, at least a fraction of $\nu$ of all examples is in the estimated region (cf. table). The large value of $\nu$ causes the additional data points in the upper left corner to have almost no influence on the decision function. For smaller values of $\nu$, such as 0.1 *(third picture)*, the points cannot be ignored anymore. Alternatively, one can force the algorithm to take these 'outliers' into account by changing the kernel width (2): in the *fourth picture*, using $c = 0.1, \nu = 0.5$, the data is effectively analyzed on a different length scale which leads the algorithm to consider the outliers as meaningful points.

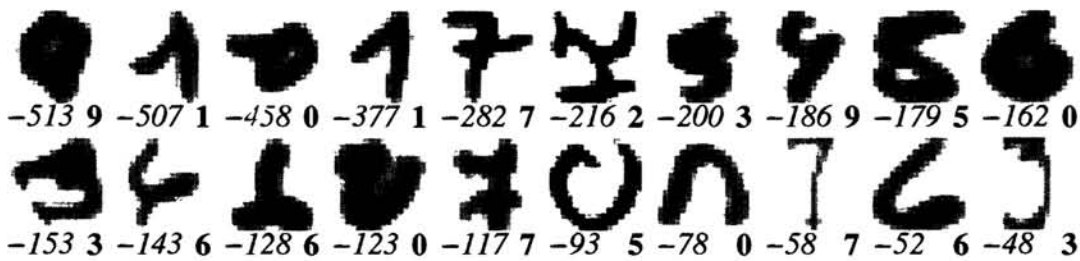

Figure 2: Outliers identified by the proposed algorithm, ranked by the negative output of the SVM (the argument of the sgn in the decision function). The outputs (for convenience in units of $10^{-5}$) are written underneath each image in italics, the (alleged) class labels are given in bold face. Note that most of the examples are "difficult" in that they are either atypical or even mislabelled.

## 5 DISCUSSION

One could view the present work as an attempt to provide an algorithm which is in line with Vapnik's principle never to solve a problem which is more general than the one that one is actually interested in. E.g., in situations where one is only interested in detecting *novelty*, it is not always necessary to estimate a full density model of the data. Indeed, density estimation is more difficult than what we are doing, in several respects.

Mathematically speaking, a density will only exist if the underlying probability measure possesses an absolutely continuous distribution function. The general problem of estimating the measure for a large class of sets, say the sets measureable in Borel's sense, is not solvable (for a discussion, see e.g. [8]). Therefore we need to restrict ourselves to making a statement about the measure of *some* sets. Given a small class of sets, the simplest estimator accomplishing this task is the empirical measure, which simply looks at how many training points fall into the region of interest. Our algorithm does the opposite. It starts with the number of training points that are supposed to fall into the region, and then estimates a region with the desired property. Often, there will be many such regions — the solution becomes unique only by applying a regularizer, which in our case enforces that the region be small in a feature space associated to the kernel. This, of course, implies, that the measure of smallness in this sense depends on the kernel used, in a way that is no different to any other method that regularizes in a feature space. A similar problem, however, appears in density estimation already when done in input space. Let $p$ denote a density on $\mathcal{X}$. If we perform a (nonlinear) coordinate transformation in the input domain $\mathcal{X}$, then the density values will *change*; loosely speaking, what remains constant is $p(x) \cdot dx$, while $dx$ is transformed, too. When directly estimating the probability *measure* of regions, we are not faced with this problem, as the regions automatically change accordingly.

An attractive property of the measure of smallness that we chose to use is that it can also be placed in the context of regularization theory, leading to an interpretation of the solution as maximally smooth in a sense which depends on the specific kernel used [3].

The main inspiration for our approach stems from the earliest work of Vapnik and collaborators. They proposed an algorithm for characterizing a set of unlabelled data points by separating it from the origin using a hyperplane [9]. However, they quickly moved on to two-class classification problems, both in terms of algorithms and in the theoretical development of statistical learning theory which originated in those days. From an algorithmic point of view, we can identify two shortcomings of the original approach which may have caused research in this direction to stop for more than three decades. Firstly, the original

algorithm in was limited to linear decision rules in input space, secondly, there was no way of dealing with outliers. In conjunction, these restrictions are indeed severe — a generic dataset need not be separable from the origin by a hyperplane in input space. The two modifications that we have incorporated dispose of these shortcomings. Firstly, the kernel trick allows for a much larger class of functions by nonlinearly mapping into a high-dimensional feature space, and thereby increases the chances of separability from the origin. In particular, using a Gaussian kernel (2), such a separation exists for any data set $x_1, \ldots, x_\ell$: to see this, note that $k(x_i, x_j) > 0$ for all $i, j$, thus all dot products are positive, implying that all mapped patterns lie inside the same orthant. Moreover, since $k(x_i, x_i) = 1$ for all $i$, they have unit length. Hence they are separable from the origin. The second modification allows for the possibility of outliers. We have incorporated this 'softness' of the decision rule using the $\nu$-trick and thus obtained a direct handle on the fraction of outliers.

We believe that our approach, proposing a concrete algorithm with well-behaved computational complexity (convex quadratic programming) for a problem that so far has mainly been studied from a theoretical point of view has abundant practical applications. To turn the algorithm into an easy-to-use black-box method for practicioners, questions like the selection of kernel parameters (such as the width of a Gaussian kernel) have to be tackled. It is our expectation that the theoretical results which we have briefly outlined in this paper will provide a foundation for this formidable task.

**Acknowledgement.**   Part of this work was supported by the ARC and the DFG (# Ja 379/9-1), and done while BS was at the Australian National University and GMD FIRST. AS is supported by a grant of the Deutsche Forschungsgemeinschaft (Sm 62/1-1). Thanks to S. Ben-David, C. Bishop, C. Schnörr, and M. Tipping for helpful discussions.

## Footnotes

[1]i.e. with a distance function that differs from a metric in that it is only semidefinite

# References

[1] S. Ben-David and M. Lindenbaum. Learning distributions by their density levels: A paradigm for learning without a teacher. *Journal of Computer and System Sciences*, 55:171–182, 1997.

[2] B. Schölkopf, C. Burges, and V. Vapnik. Extracting support data for a given task. In U. M. Fayyad and R. Uthurusamy, editors, *Proceedings, First International Conference on Knowledge Discovery & Data Mining*. AAAI Press, Menlo Park, CA, 1995.

[3] B. Schölkopf, J. Platt, J. Shawe-Taylor, A.J. Smola, and R.C. Williamson. Estimating the support of a high-dimensional distribution. TR MSR 99 - 87, Microsoft Research, Redmond, WA, 1999.

[4] B. Schölkopf, A. Smola, and K.-R. Müller. Kernel principal component analysis. In B. Schölkopf, C. Burges, and A. Smola, editors, *Advances in Kernel Methods — Support Vector Learning*. MIT Press, Cambridge, MA, 1999. 327 – 352.

[5] B. Schölkopf, R. Williamson, A. Smola, and J. Shawe-Taylor. Single-class support vector machines. In J. Buhmann, W. Maass, H. Ritter, and N. Tishby, editors, *Unsupervised Learning*, Dagstuhl-Seminar-Report 235, pages 19 – 20, 1999.

[6] A. Smola, R. C. Williamson, S. Mika, and B. Schölkopf. Regularized principal manifolds. In *Computational Learning Theory: 4th European Conference*, volume 1572 of *Lecture Notes in Artificial Intelligence*, pages 214 – 229. Springer, 1999.

[7] D.M.J. Tax and R.P.W. Duin. Data domain description by support vectors. In M. Verleysen, editor, *Proceedings ESANN*, pages 251 – 256, Brussels, 1999. D Facto.

[8] V. Vapnik. *Statistical Learning Theory*. Wiley, New York, 1998.

[9] V. Vapnik and A. Lerner. Pattern recognition using generalized portraits. *Avtomatika i Telemekhanika*, 24:774 – 780, 1963.
